# Iterative scaled trust-region learning in Krylov subspaces via Pearlmutter's implicit sparse Hessian-vector multiply

**Eiji Mizutani**
Department of Computer Science
Tsing Hua University
Hsinchu, 300 TAIWAN R.O.C.
eiji@wayne.cs.nthu.edu.tw

**James W. Demmel**
Mathematics and Computer Science
University of California at Berkeley,
Berkeley, CA 94720 USA
demmel@cs.berkeley.edu

## Abstract

The online incremental gradient (or backpropagation) algorithm is widely considered to be the fastest method for solving large-scale neural-network (NN) learning problems. In contrast, we show that an appropriately implemented *iterative batch-mode* (or *block-mode*) *learning* method can be much faster. For example, it is three times faster in the UCI letter classification problem (26 outputs, 16,000 data items, 6,066 parameters with a two-hidden-layer multilayer perceptron) and 353 times faster in a nonlinear regression problem arising in color recipe prediction (10 outputs, 1,000 data items, 2,210 parameters with a neuro-fuzzy modular network). The three principal innovative ingredients in our algorithm are the following: First, we use *scaled trust-region regularization* with inner-outer iteration to solve the associated "overdetermined" nonlinear least squares problem, where the inner iteration performs a truncated (or inexact) Newton method. Second, we employ Pearlmutter's implicit sparse Hessian matrix-vector multiply algorithm to construct the Krylov subspaces used to solve for the truncated Newton update. Third, we exploit sparsity (for preconditioning) in the matrices resulting from the NNs having many outputs.

## 1 Introduction

Our objective function to be minimized for optimizing the $n$-dimensional parameter vector $\boldsymbol{\theta}$ of an $F$-output NN model is the sum over all the $d$ data of squared residuals: $E(\boldsymbol{\theta}) = \frac{1}{2}\|\mathbf{r}(\boldsymbol{\theta})\|_2^2 = \frac{1}{2}\sum_{i=1}^{m} r_i^2 = \frac{1}{2}\sum_{k=1}^{F} \|\mathbf{r}_k\|_2^2$. Here, $m \equiv Fd$; $\mathbf{r}(\boldsymbol{\theta})$ is the $m$-dimensional residual vector composed of all $m$ residual elements: $r_i$ ($i = 1, \ldots, m$); and $\mathbf{r}_k$ the $d$-dimensional residual vector evaluated at terminal node $k$. The gradient vector and the Hessian matrix of $E(\boldsymbol{\theta})$ are given by $\mathbf{g} \equiv \mathbf{J}^T\mathbf{r}$ and $\mathbf{H} \equiv \mathbf{J}^T\mathbf{J} + \mathbf{S}$, respectively, where $\mathbf{J}$, the $m \times n$ (**residual**) **Jacobian matrix** of $\mathbf{r}$, is readily obtainable from *backpropagation* (BP) process, and $\mathbf{S}$ is the matrix of second-derivative

terms of $\mathbf{r}$; i.e., $\mathbf{S} \equiv \sum_{i=1}^{m} r_i \nabla^2 r_i$. Most nonlinear least squares algorithms take advantage of information of $\mathbf{J}$ or its cross product called the **Gauss-Newton** (GN) Hessian $\mathbf{J}^T\mathbf{J}$ (or the *Fisher information matrix* for $E(.)$ in Amari's natural-gradient learning [1]), which is the important portion of $\mathbf{H}$ because influence of $\mathbf{S}$ becomes weaker and weaker as residuals become smaller while learning progresses. With multiple $F$-output nonlinear models (except *fully-connected* NNs), $\mathbf{J}$ is known to have the $m \times n$ **block angular** matrix form (see [7, 6] and references therein). For instance, consider a single-hidden layer $S$-$H$-$F$ MLP (with $S$-input $H$-hidden $F$-output nodes); there are $n_A = F(H + 1)$ **terminal parameters** $\boldsymbol{\theta}^A$ (including threshold parameters) on direct connections to $F$ terminal nodes, each of which has $C_A(=H + 1)$ direct connections, and the rest of $n_B = H(S + 1)$ parameters are not directly connected to any terminal node; hence, $n_B$ **hidden parameters** $\boldsymbol{\theta}^B$. In other words, model's parameters $\boldsymbol{\theta}$ ($n = FC_A + n_B$ in total) can separate as: $\boldsymbol{\theta}^T = [\boldsymbol{\theta}^{A^T} | \boldsymbol{\theta}^{B^T}] = [\boldsymbol{\theta}_1^{A^T}, \cdots, \boldsymbol{\theta}_k^{A^T}, \cdots, \boldsymbol{\theta}_F^{A^T} | \boldsymbol{\theta}^{B^T}]$, where $\boldsymbol{\theta}_k^A$ is a vector of the $k$th subset of $C_A$ terminal parameters directly linked to terminal node $k$ ($k = 1, \cdots, F$). The associated residual Jacobian matrix $\mathbf{J}$ can be given in the block-angular form below left, and thus the (full) Hessian matrix $\mathbf{H}$ has the $n \times n$ *sparse* **block arrow** form below right ($\times$ denotes some non-zero block) as well as the GN-Hessian $\mathbf{J}^T\mathbf{J}$:

$$\underbrace{\mathbf{J}}_{m \times n} = \begin{bmatrix} \mathbf{A}_1 & & & & \mathbf{B}_1 \\ & \mathbf{A}_2 & & & \mathbf{B}_2 \\ & & \ddots & & \vdots \\ & & & \mathbf{A}_F & \mathbf{B}_F \end{bmatrix}, \qquad \underbrace{\mathbf{H}}_{n \times n} = \begin{bmatrix} \times & & & & \times \\ & \times & & & \times \\ & & \times & & \times \\ & & & \times & \times \\ \times & \times & \times & \times & \times \end{bmatrix}. \quad (1)$$

Here in $\mathbf{J}$, $\mathbf{A}_k$ and $\mathbf{B}_k$ are $d \times C_A$ and $d \times n_B$ Jacobian matrices, respectively, of the $d$-dimensional residual vector $\mathbf{r}_k$ evaluated at terminal node $k$. Notice that there are $F$ diagonal $\mathbf{A}_k$ blocks [because $(F - 1)C_A$ terminal parameters excluding $\boldsymbol{\theta}_k^A$ have no effect on $\mathbf{r}_k$], and $F$ vertical $\mathbf{B}_k$ blocks corresponding to the $n_B$ hidden parameters $\boldsymbol{\theta}^B$ that contribute to minimizing all the residuals $\mathbf{r}_k(k=1, \cdots, F)$ evaluated at all $F$ terminal nodes. Therefore, the posed problem is *overdetermined* when "$m > n$" (namely, "$d > C_A + \frac{1}{F}n_B$") holds. In addition, when the terminal nodes have *linear identity functions*, terminal parameters $\boldsymbol{\theta}^A$ are linear, and thus all $\mathbf{A}_k$ blocks become identical $\mathbf{A}_1 = \mathbf{A}_2 = \cdots = \mathbf{A}_F$, with $H + 1$ hidden-node outputs (including one *constant* bias-node output) in each row. For small- and medium-scale problems, *direct batch-mode learning* is recommendable with a suitable "direct" matrix factorization, but attention must be paid to exploiting obvious *sparsity* in either *block-angular* $\mathbf{J}$ or *block-arrow* $\mathbf{H}$ so as to render the algorithms efficient in both memory and operation counts [7, 6]. Notice that $\mathbf{H}^{-1}$ is *dense* even if $\mathbf{H}$ has a nice block-arrow sparsity structure. For large-scale problems, Krylov subspace methods, which circumvent the need to perform time-consuming and memory-intensive direct matrix factorizations, can be employed to realize what we call **iterative batch-mode learning.** If any rows (or columns) of those matrices $\mathbf{A}_k$ and $\mathbf{B}_k$ are not needed explicitly, then Pearlmutter's method [11] can automatically exploit such sparsity to perform sparse Hessian-vector product in constructing a Krylov subspace for parameter optimization, which we describe in what follows with our numerical evidence.

## 2  Inner-Outer Iterative Scaled Trust-Region Methods

*Practical Newton methods* enjoy both the global convergence property of the Cauchy (or steepest descent) method and the fast local convergence of the Newton method.

## 2.1 Outer iteration process in trust-region methods

One might consider a convex combination of the Cauchy step $\Delta\boldsymbol{\theta}_{Cauchy}$ and the Newton step $\Delta\boldsymbol{\theta}_{Newton}$ such as (using a scalar parameter $h$):

$$\Delta\boldsymbol{\theta}_{Dogleg} \stackrel{\text{def}}{=} (1-h)\Delta\boldsymbol{\theta}_{Cauchy} + h\Delta\boldsymbol{\theta}_{Newton}, \tag{2}$$

which is known as the *dogleg step* [4, 9]. This step yields a good approximate solution to the so-called "scaled 2-norm" or "$M$-norm" **trust-region subproblem** (e.g., see Chap. 7 in [2]) with *Lagrange multiplier* $\mu$ below:

$$\min_{\Delta\boldsymbol{\theta}} q(\Delta\boldsymbol{\theta}) \text{ subject to } \|\Delta\boldsymbol{\theta}\|_M \leq R, \text{ or } \min_{\Delta\boldsymbol{\theta}} \left\{ q(\Delta\boldsymbol{\theta}) + \tfrac{\mu}{2}(\Delta\boldsymbol{\theta}^T \mathbf{M}\Delta\boldsymbol{\theta} - R^2) \right\}, \tag{3}$$

where the distances are measured in the $M$-norm: $\|\mathbf{x}\|_M = \sqrt{\mathbf{x}^T\mathbf{M}\mathbf{x}}$ with a symmetric positive definite matrix $\mathbf{M}$, and $R$ (called the *trust-region radius*) signifies the trust-region size of the local quadratic model $q(\Delta\boldsymbol{\theta}) \stackrel{\text{def}}{=} E(\boldsymbol{\theta}) + \mathbf{g}^T\Delta\boldsymbol{\theta} + \tfrac{1}{2}\Delta\boldsymbol{\theta}^T\mathbf{H}\Delta\boldsymbol{\theta}$. Radius $R$ is controlled according to how well $q(.)$ predicts the behavior of $E(.)$ by checking the *error reduction ratio* below:

$$\rho = \frac{\text{Actual error reduction}}{\text{Predicted error reduction}} = \frac{E(\boldsymbol{\theta}_{\text{now}}) - E(\boldsymbol{\theta}_{\text{next}})}{E(\boldsymbol{\theta}_{\text{now}}) - q(\Delta\boldsymbol{\theta})}. \tag{4}$$

For more details, refer to [9, 2]. The posed constrained quadratic minimization can be solved with *Lagrange multiplier* $\mu$: If $\Delta\boldsymbol{\theta}$ is a solution to the posed problem, then $\Delta\boldsymbol{\theta}$ satisfies the formula: $(\mathbf{H} + \mu\mathbf{M})\Delta\boldsymbol{\theta} = -\mathbf{g}$, with $\mu(\|\Delta\boldsymbol{\theta}\|_M - R) = 0$, $\mu \geq 0$, and $\mathbf{H} + \mu\mathbf{M}$ positive semidefinite. In nonlinear least squares context, the nonnegative scalar parameter $\mu$ is known as the **Levenberg-Marquardt parameter**. When $\mu = 0$ (namely, $R \geq \|\Delta\boldsymbol{\theta}_{Newton}\|_M$), the trust-region step $\Delta\boldsymbol{\theta}$ becomes the Newton step $\Delta\boldsymbol{\theta}_{Newton} \stackrel{\text{def}}{=} -\mathbf{H}^{-1}\mathbf{g}$, and, as $\mu$ increases (i.e., as $R$ decreases), $\Delta\boldsymbol{\theta}$ gets closer to the (full) Cauchy step $\Delta\boldsymbol{\theta}_{Cauchy}$: $\Delta\boldsymbol{\theta}_{Cauchy} \stackrel{\text{def}}{=} -\left(\mathbf{g}^T\mathbf{M}^{-1}\mathbf{g}/\mathbf{g}^T\mathbf{M}^{-1}\mathbf{H}\mathbf{M}^{-1}\mathbf{g}\right)\mathbf{M}^{-1}\mathbf{g}$. When $R < \|\Delta\boldsymbol{\theta}_{Cauchy}\|_M$, the trust-region step $\Delta\boldsymbol{\theta}$ reduces to the *restricted* Cauchy step $\Delta\boldsymbol{\theta}_{RC} \stackrel{\text{def}}{=} -(R/\|\Delta\boldsymbol{\theta}_{Cauchy}\|_M)\Delta\boldsymbol{\theta}_{Cauchy}$. If $\|\Delta\boldsymbol{\theta}_{Cauchy}\|_M < R < \|\Delta\boldsymbol{\theta}_{Newton}\|_M$, $\Delta\boldsymbol{\theta}$ is the "dogleg step," *intermediate* between $\Delta\boldsymbol{\theta}_{Cauchy}$ and $\Delta\boldsymbol{\theta}_{Newton}$, as shown in Eq. (2), where scalar $h$ $(0 < h < 1)$ is the positive root of $\|\mathbf{s} + h\mathbf{p}\|_M = R$:

$$h = \frac{-\mathbf{s}^T\mathbf{M}\mathbf{p} + \sqrt{(\mathbf{s}^T\mathbf{M}\mathbf{p})^2 + \mathbf{p}^T\mathbf{M}\mathbf{p}(R^2 - \mathbf{s}^T\mathbf{M}\mathbf{s})}}{\mathbf{p}^T\mathbf{M}\mathbf{p}}, \tag{5}$$

with $\mathbf{s} \stackrel{\text{def}}{=} \Delta\boldsymbol{\theta}_{Cauchy}$ and $\mathbf{p} \stackrel{\text{def}}{=} \Delta\boldsymbol{\theta}_{Newton} - \Delta\boldsymbol{\theta}_{Cauchy}$ (when $\mathbf{p}^T\mathbf{g} < 0$). In this way, the trial step $\Delta\boldsymbol{\theta}$ is subject to *trust-region regularization.*

In large-scale problems, the linear-equation solution sequence $\{\Delta\boldsymbol{\theta}_k\}$ is generated iteratively while seeking a trial step $\Delta\boldsymbol{\theta}$ in the *inner iteration* process, and the parameter sequence $\{\boldsymbol{\theta}_i\}$, whose two consecutive elements are denoted by $\boldsymbol{\theta}_{\text{now}}$ and $\boldsymbol{\theta}_{\text{next}}$, is produced by the *outer iteration* (i.e., *epoch* in batch mode). The outer iterative process updates parameters by $\boldsymbol{\theta}_{\text{next}} = \boldsymbol{\theta}_{\text{now}} + \Delta\boldsymbol{\theta}$ without taking any *uphill* movement: That is, if the step is not satisfactory, then $R$ is decreased so as to realize an important *Levenberg-Marquardt* concept: *the failed step is shortened and deflected towards the Cauchy-step direction simultaneously.* For this purpose, the trust-region methods compute the gradient vector in batch mode or with (sufficiently large) data block (i.e., *block mode*; see our demonstration in Section 3).

## 2.2 Inner iteration process with truncated preconditioned linear CG

We employ a preconditioned conjugate gradient (PCG) (among many Krylov subspace methods; see Section 6.6 in [3] and Chapter 5 in [2]) with our symmetric

positive definite preconditioner $\mathbf{M}$ for solving the $M$-norm trust-region subproblem (3). This is the *truncated* PCG (also known as *Steihaug-Toint CG*) applicable even to *nonconvex* problems for solving *inexactly* the Newton formula by the *inner iterative process* below (see pp. 628-629 in [10]; pp. 202–218 in [2]) based on the standard PCG algorithm (e.g., see page 317 in [3]):

**Algorithm 1**: *The inner iteration process via preconditioned CG.*
1. Initialization ($k{=}0$):
   Set $\Delta\boldsymbol{\theta}_0 = \mathbf{0}$ and $\boldsymbol{\delta}_0 = -\mathbf{g}$ $(=-\mathbf{g} - \mathbf{H}\Delta\boldsymbol{\theta}_0)$;
   Solve $\mathbf{Mz} = \boldsymbol{\delta}_0$ for pseudoresiduals: $\mathbf{z} = \mathbf{M}^{-1}\boldsymbol{\delta}_0$;
   Compute $\tau_0 = \boldsymbol{\delta}_0^T \mathbf{z}$;
   Set $k = 1$ and $\mathbf{d}_1 = \mathbf{z}$, and then proceed to Step 2.
2. Matrix-vector product: $\mathbf{z} = \mathbf{Hd}_k = \mathbf{J}^T(\mathbf{Jd}_k) + \mathbf{Sd}_k$   (see also **Algorithm 2**).
3. Curvature check: $\gamma_k = \mathbf{d}_k^T \mathbf{z} = \mathbf{d}_k^T \mathbf{Hd}_k$.
   If $\gamma_k > 0$, then continue with Step 4. Otherwise, compute $h$ $(>0)$ such that
   $\|\Delta\boldsymbol{\theta}_{k-1} + h\mathbf{d}_k\|_M = R$, and **terminate** with $\Delta\boldsymbol{\theta} = \Delta\boldsymbol{\theta}_{k-1} + h\mathbf{d}_k$.
4. Step size: $\eta_k = \frac{\tau_{k-1}}{\gamma_k}$.
5. Approximate solution: $\Delta\boldsymbol{\theta}_k = \Delta\boldsymbol{\theta}_{k-1} + \eta_k\mathbf{d}_k$.
   If $\|\Delta\boldsymbol{\theta}_k\|_M < R$, go onto Step 6; else **terminate** with $\quad \Delta\boldsymbol{\theta} = \frac{R}{\|\Delta\boldsymbol{\theta}_k\|_M}\Delta\boldsymbol{\theta}_k.$   (6)
6. Linear-system residuals: $\boldsymbol{\delta}_k = \boldsymbol{\delta}_{k-1} - \eta_k\mathbf{z}$ $[= -\mathbf{g} - \mathbf{H}\Delta\boldsymbol{\theta}_k = -q\,'(\Delta\boldsymbol{\theta}_k)]$.
   If $\|\boldsymbol{\delta}_k\|_2$ is small enough; i.e., $\|\boldsymbol{\delta}_k\|_2 \leq \xi\|\mathbf{g}\|_2$, then **terminate** with $\Delta\boldsymbol{\theta} = \Delta\boldsymbol{\theta}_k$.
7. Pseudoresiduals: $\mathbf{z} = \mathbf{M}^{-1}\boldsymbol{\delta}_k$, and then compute $\tau_k = \boldsymbol{\delta}_k^T\mathbf{z}$.
8. Conjugation factor: $\beta_{k+1} = \frac{\tau_k}{\tau_{k-1}}$.
9. Search direction: $\mathbf{d}_{k+1} = \boldsymbol{\delta}_k + \beta_{k+1}\mathbf{d}_k$.
10. If $k < k_{\text{limit}}$, set $k = k + 1$ and return to Step 2.
    Otherwise, **terminate** with $\Delta\boldsymbol{\theta} = \Delta\boldsymbol{\theta}_k$. $\square$

At Step 3, $h$ is obtainable with Eq. (5) with $\mathbf{s} = \Delta\boldsymbol{\theta}_{k-1}$ and $\mathbf{p} = \mathbf{d}_k$ plugged in. Likewise, in place of Eq. (6) at Step 5, we may use Eq. (5) for $\Delta\boldsymbol{\theta} = \Delta\boldsymbol{\theta}_{k-1} + h\mathbf{d}_k$ such that $\|\Delta\boldsymbol{\theta}_{k-1} + h\mathbf{d}_k\|_M = R$, but both computations become identical if $R \leq \|\Delta\boldsymbol{\theta}_{Cauchy}\|_M$; otherwise, Eq. (6) is less expensive and tends to give more bias towards the Newton direction. The inner-iterative process terminates (i.e., stops at inner iteration $k$) when one of the next four conditions holds:

(A) $\mathbf{d}_k^T\mathbf{Hd}_k \leq 0$,  (B) $\|\Delta\boldsymbol{\theta}_k\|_M \geq R$,  (C) $\|\mathbf{H}\Delta\boldsymbol{\theta}_k + \mathbf{g}\|_2 \leq \xi\|\mathbf{g}\|_2$,  (D) $k{=}k_{\text{limit}}$.   (7)

Condition (D) at Step 10 is least likely to be met since there would be no prior knowledge about preset limits $k_{\text{limit}}$ to inner iterations (usually, $k_{\text{limit}}{=}n$). As long as $\mathbf{d}_k^T\mathbf{Hd}_k > 0$ holds, PCG works properly until the CG-trajectory hits the trust-region boundary [Condition (B) at Step 5], or till the 2-norm linear-system residuals become small [Condition (C) at Step 6], where $\xi$ can be fixed (e.g., $\xi{=}0.01$). Condition (A) $\mathbf{d}_k^T\mathbf{Hd}_k \leq 0$ (at Step 3) may hold when the local model is not strictly convex (or $\mathbf{H}$ is not positive definite). That is, $\mathbf{d}_k$ is a *direction of zero or negative curvature*; a typical exploitation of non-positive curvature is to set $\Delta\boldsymbol{\theta}$ equal to the "step to the trust-region boundary along that curvature segment (in Step 3)" as a model minimizer in the trust region. In this way, the terminated $k$th CG step yields an approximate solution to the trust-region subproblem (3), and it belongs to the Krylov subspace $span\{-\mathbf{M}^{-\frac{1}{2}}\mathbf{g}, -(\mathbf{M}^{-\frac{1}{2}}\mathbf{HM}^{-\frac{1}{2}})\mathbf{M}^{-\frac{1}{2}}\mathbf{g}, ..., -(\mathbf{M}^{-\frac{1}{2}}\mathbf{HM}^{-\frac{1}{2}})^{k-1}\mathbf{M}^{-\frac{1}{2}}\mathbf{g}\}$, resulting from our application of CG (without multiplying by $\mathbf{M}^{-\frac{1}{2}}$) to the *symmetric* Newton formula $(\mathbf{M}^{-\frac{1}{2}}\mathbf{HM}^{-\frac{1}{2}})(\mathbf{M}^{\frac{1}{2}}\Delta\boldsymbol{\theta}) = -\mathbf{M}^{-\frac{1}{2}}\mathbf{g}$, because $\mathbf{M}^{-1}\mathbf{H}$ (in the system $\mathbf{M}^{-1}\mathbf{H}\Delta\boldsymbol{\theta} = -\mathbf{M}^{-1}\mathbf{g}$) is unlikely symmetric (see page 317 in [3]) even if $\mathbf{M}$ is a diagonal matrix (unless $\mathbf{M} = \mathbf{I}$).

The overall memory requirement of Algorithm 1 is $O(n)$ because at most five $n$-vectors are enough to implement. Since the matrix-vector product $\mathbf{H}\mathbf{d}_k$ at Step 2 is dominant in operation cost of the entire inner-outer process, we can employ Pearlmutter's method with no $\mathbf{H}$ explicitly required. To better understand the method, we first describe a straightforward implicit sparse matrix-vector multiply when $\mathbf{H} = \mathbf{J}^T\mathbf{J}$; it evaluates $\mathbf{J}^T\mathbf{J}\mathbf{d}_i$ (without forming $\mathbf{J}^T\mathbf{J}$) in two-step *implicit matrix-vector product* as $\mathbf{z}=\mathbf{J}^T(\mathbf{J}\mathbf{d}_i)$, exploiting block-angular $\mathbf{J}$ in Eq. (1); i.e., working on each block, $\mathbf{A}_k$ and $\mathbf{B}_k$, in a row-wise manner below:

**Algorithm 2**: *Implicit (i.e., matrix-free) sparse matrix-vector multiplication step with an $F$-output NN model at inner iteration $i$ starting with* $\mathbf{z} = \mathbf{0}$:

  *for $p = 1$ to $d$* (i.e., one sweep of $d$ training data):

      (a) do forward pass to compute $F$ final outputs $\mathbf{y}_p(\boldsymbol{\theta})$ on datum $p$;

    *for $k = 1$ to $F$* (at each terminal node $k$):

        • (b) do backward pass to obtain the $p$th row of $\mathbf{A}_k$ as the $C_A$-vector $\mathbf{a}_{p,k}^T$, and the $p$th row of $\mathbf{B}_k$ as the $n_B$-vector $\mathbf{b}_{p,k}^T$;

        • (c) compute $\alpha_k\mathbf{a}_p$ and $\alpha_k\mathbf{b}_{p,k}$, where scalar $\alpha_k = \mathbf{a}_{p,k}^T\mathbf{d}_{i,k}^a + \mathbf{b}_{p,k}^T\mathbf{d}_i^b$, and then add them to their corresponding elements of $\mathbf{z}$;

    *end for $k$.*

  *end for $p$.* □

Here, Step (a) costs *at least $2dn$* (see details in [8]); Step (b) costs *at least $2ml_u$*, where $m=Fd$ and $l_u=C_A+n_B < n=FC_A+n_B$; and Step (c) costs $4ml_u$; overall, Algorithm 2 costs $O(ml_u)$, *linear* in $F$. Note that if sparsity is ignored, the cost becomes $O(mn)$, *quadratic* in $F$ since $mn = Fd(FC_A+n_B)$. Algorithm 2 can extract explicitly $F$ pairs of row vectors ($\mathbf{a}^T$ and $\mathbf{b}^T$) of $\mathbf{J}$ (with $Fl_u$ storage) on each datum, making it easier to apply other numerical linear algebra approaches such as *preconditioning* to reduce the number of inner iterations. Yet, if the row vectors are not needed explicitly, then Pearlmutter's method is more efficient, calculating $\alpha_k$ [see Step (c)] in its forward pass (i.e., $R\{y_k\}=\alpha_k$; see Eq. (4.3) on page 151 in [11]). When $\mathbf{H} = \mathbf{J}^T\mathbf{J}$, it is easy to simplify its backward pass (see Eq. (4.4) on page 152 in [11]), just by eliminating the terms involving residuals $r$ and second-derivatives of node functions $f''(.)$, so as to multiply vectors $\mathbf{a}_k$ and $\mathbf{b}_k$ through by scalar $\alpha_k$ *implicitly*. This simplified method of Pearlmutter runs in time $O(dn)$, whereas Algorithm 2 does in $O(ml_u)$. Since $ml_u - dn = dF(C_A + n_B) - d(FC_A + n_B) = d(F-1)n_B$, Pearlmutter's method can be up to $F$ times faster than Algorithm 2. Furthermore, Pearlmutter's original method efficiently multiplies an $n$-vector by the "full" Hessian matrix still in $O(dn)$ for $\mathbf{z} = \mathbf{H}\mathbf{d}_i = \mathbf{J}^T(\mathbf{J}\mathbf{d}_i) + \mathbf{S}\mathbf{d}_i = \sum_{j=1}^m (\mathbf{u}_j^T\mathbf{d}_i)\mathbf{u}_j + \sum_{j=1}^m [\nabla^2 r_j]r_j\mathbf{d}_i$, where $\mathbf{u}_i^T$ is the $i$th row vector of $\mathbf{J}$; notably, the method automatically exploits block-arrow sparsity of $\mathbf{H}$ [see Eq. (1), right] in the essentially same way as the standard BP deals with block-angular sparsity of $\mathbf{J}$ [see Eq. (1), left] to perform the matrix-vector product $\mathbf{g} = \mathbf{J}^T\mathbf{r}$ in $O(dn)$.

## 3   Experiments and Discussion

In simulation, we compared the following five algorithms:

**Algorithm A**: Online-BP (i.e., $\mathbf{H} = \mathbf{I}$) with a fixed momentum (0.8);

**Algorithm B**: Algorithm 2 alone for Algorithm 1 with $\mathbf{H} = \mathbf{J}^T\mathbf{J}$ (see [6]);

**Algorithm C**: Pearlmutter's method alone for Algorithm 1 with $\mathbf{H} = \mathbf{J}^T\mathbf{J}$;

**Algorithm D**: Algorithm 2 to obtain preconditioner $\mathbf{M} = \mathrm{diag}(\mathbf{J}^T\mathbf{J})$ only, and Pearlmutter's method for Algorithm 1 with $\mathbf{H} = \mathbf{J}^T\mathbf{J}$;

**Algorithm E**: Same as Algorithm D except with "full" Hessian $\mathbf{H} = \mathbf{J}^T\mathbf{J} + \mathbf{S}$. □

Algorithm A is tested for our speed comparison purpose, because if it works, it's probably fastest. In Algorithms D and E, Algorithm 2 was only employed for obtaining a diagonal preconditioner $\mathbf{M} = \mathrm{diag}(\mathbf{J}^T\mathbf{J})$ (or *Jacobi preconditioner*) for Algorithm 1, whereas in Algorithms B and C, no preconditioning ($\mathbf{M} = \mathbf{I}$) was applied. The performance comparisons were made with a nonlinear regression task and a classification benchmark, the letter recognition problem, from the UCI machine learning repository. All the experiments were conducted on a 1.6-GHz Pentium-IV PC with FreeBSD 4.5 and gcc-2.95.3 compiler (with -O2 optimization flag).

The first regression task was a real-world application *color recipe prediction*, a problem of determining mixing proportions of available colorants to reproduce a given target color, requiring mappings from 16 inputs (16 spectral reflectance signals of the target color) to *ten outputs* ($F$=10) (ten colorant proportions) using 1,000 training data ($d$=1,000; $m$=10,000) with 302 test data. The table below shows the results averaged over 20 trials with a single 16-82-10 MLP [$n$=2,224 ($C_A$=83;$n_B$=1,394;$l_u$=1,477); hence, $\frac{ml_u}{dn}$=6.6], which was optimized until "training RMSE $\leq$ 0.002 (application requirement)" satisfied, when we say that "convergence" (relatively early stop) occurs. Clearly, the posed regression task is nontrivial because Algorithm A, online-BP, took roughly six days (averaged over only ten trials), nearly 280 (=8748.4/31.2) times slower than (fastest) Algorithm D. In *generalization* performance, all the posed algorithms were more or less equivalent.

| Model | Single 16-82-10 MLP | | | | | Five-MLP mixed | | |
|---|---|---|---|---|---|---|---|---|
| Algorithm | A | B | C | D | E | B | C | D |
| Total time (min) | 8748.4 | 336.4 | 107.2 | 31.2 | 64.5 | 162.3 | 57.6 | 20.9 |
| Stopped epoch | 2,916,495.2 | 272.5 | 261.5 | 132.7 | 300.3 | 147.3 | 160.0 | 179.1 |
| Time/epoch (sec) | 0.2 | 73.8 | 24.6 | 14.1 | 12.9 | 65.2 | 21.6 | 7.0 |
| Inner itr./epoch | N/A | 218.3 | 216.0 | 142.7 | 110.9 | 193.8 | 174.1 | 66.0 |
| Flops ratio/itr. | N/A | 3.9 | 1.0 | | 1.3 | 4.1 | 1.2 | |
| Test RMSE | 0.020 | 0.015 | 0.015 | 0.015 | 0.015 | 0.016 | 0.016 | 0.017 |

We also observed that use of full Hessian matrix (Algorithm E) helped reduce inner iterations per epoch, although the total convergence time turned out to be greater than that obtained with the GN-Hessian (Algorithm D), presumably because our Jacobi-preconditioner must be more suitable for the GN-Hessian than for the full Hessian, and perhaps because the inner iterative process of Algorithm E can terminate due to detection of non-positive curvature in Eq. (7)(A); this extra chance of termination may increase the total epochs, but help reduce the time per epoch. Remarkably, the time per inner iteration of Algorithm E did not differ much from Algorithms C and D owing to Pearlmutter's method; in fact, given preconditioner $\mathbf{M}$, Algorithm E merely needed about 1.3 times more *flops* * per inner iteration than Algorithms C and D did, although Algorithm B needed nearly 3.9 times more. The measured *megaflop rates* for all these codes lie roughly in the range from 200-270 Mflop/sec; typically, below 10 % of peak machine speed.

For improving single-MLP performance, one might employ *two* layers of hidden nodes (rather than *one large* hidden layer; see the letter problem below), which increases $n_B$ while reducing $n_A$, rendering Algorithm 2 less efficient (i.e., slower). Alternatively, one might introduce *direct connections* between the input and terminal output layers, which increases $C_A$, the column size of $\mathbf{A}_k$, retaining nice parameter separability. Yet another approach (if applicable) is to use a "comple-

mentary mixtures of $Z$ MLP-experts" model (or a neuro-fuzzy modular network) that combines $Z$ smaller-size MLPs *complementarily*; the associated residual vector to be minimized becomes:    $\mathbf{r}(\boldsymbol{\theta}) = \mathbf{y}(\boldsymbol{\theta}) - \mathbf{t} = \left[\sum_{i=1}^{Z} w_i \mathbf{o}_i\right] - \mathbf{t}$,  where scalar $w_i$, the $i$th output of the integrating unit, is the $i$th (normalized) mixing proportion assigned to the outputs ($F$-vector $\mathbf{o}_i$) of expert-MLP $i$. Note that each expert learns "residuals" rather than "desired outputs" (unlike in the committee method below) in the sense that only the final combined outputs $\mathbf{y}$ must come close to the desired ones $\mathbf{t}$. That is, there are strong coupling effects (see page 80 in [5]) among all experts; hence, it is crucial to consider the *global* Hessian across all experts to optimize them *simultaneously* [7]. The corresponding $\mathbf{J}$ has the same block-angular form as that in Eq. (1)(left) with $\mathbf{A}_k \equiv [\mathbf{A}_k^1 \mathbf{A}_k^2 \cdots \mathbf{A}_k^Z]$, and $\mathbf{B}_k \equiv [\mathbf{B}_k^1 \mathbf{B}_k^2 \cdots \mathbf{B}_k^Z]$ ($k = 1, \cdots, F$). Here, the residual Jacobian portion for the parameters of the integrating unit was omitted because they were merely fine-tuned with a *steepest-descent* type method owing to our knowledge-based design for input-partition to avoid (too many) local experts. Specifically, the spectral reflectance signals (16 inputs) were converted to the *hue angle* as input to the integrating unit that consists of five bell-shaped basis functions, partitioning that hue-subspace alone in a fuzzy fashion into only five color regions (red, yellow, green, blue, and violet) for five 16-16-10 MLP-experts, each of which receives all the 16 spectral signals as input [hence, $Z$=5; $n$=2,210 ($C_A$=85; $n_B$=1,360); $\frac{ml_u}{dn}$=6.5]. Due to localized parameter-tunings, our five-MLP mixtures model was better in learning; see faster learning in table above. In particular, our model with Algorithm D worked 353 ($\approx 123.1 \times 60.0/20.9$) times faster than with Algorithm A that took 123.1 hours (see [6]) and 419 ($\approx 8748.4/20.9$) times faster than the single MLP with Algorithm A. For our complementary mixtures model, $R\{.\}$-operator of Pearlmutter's method is readily applicable; for instance, at terminal node $k$ ($k$=1,$\cdots$,$F$):    $R\{r_k\} = R\{y_k\} = \sum_i^Z R\{o_{i,k}\}w_i + \sum_i^Z R\{w_i\}o_{i,k}$, where each $R\{o_{i,k}\}$ yields $\alpha_k$ [see Algorithm 2(c)] for each expert-MLP $i$ ($i = 1, \cdots, Z$).

The second letter classification benchmark problem involves 16 inputs (features) and 26 outputs (alphabets) with 16,000 training data ($F$=26; $d$=16,000; $m$=416,000) plus 4,000 test data. We used the 16-70-50-26 MLP (see [12]) ($n$=6,066) with 10 sets of different initial parameters randomly generated uniformly in the range $[-0.2, 0.2]$. We implemented **block-mode learning** (as well as batch mode) just by splitting the training data set into two or four equally-sized data blocks, and each data block alone is employed for Algorithms 1 and 2 except for computing $\rho$ in Eq. (4), where evaluation of $E(.)$ involves all the $d$ training data. Notice that two-block mode learning scheme updates model's parameters $\boldsymbol{\theta}$ twice per epoch, whereas online-BP updates them on each datum (i.e., $d$ times per epoch). We observed that possible *redundancy* in the data set appeared to help reduce the number of inner iterations, speeding up our *iterative batch-mode learning*; therefore, we did not use preconditioning. The next table shows the average performance (over ten trials) when the best test-set performance was obtained by epoch 1,000 with online-BP (i.e., Algorithm A) and by epoch 50 with Algorithm C in three learning modes:

| *Average* results | Online-BP | Four-block mode | Two-block mode | Batch mode |
|---|---|---|---|---|
| Total time (min) | 63.2 | 22.4 | 41.0 | 61.1 |
| Stopped epoch | 597.8 | 36.6 | 22.1 | 27.1 |
| Time/epoch (sec) | 6.3 | 36.8 | 111.7 | 135.2 |
| Avg. inner itr. | N/A | 4.5/block | 26.3/block | 31.0/batch |
| Error (train/test) | 2.3% / 6.4% | 2.7% / 5.1% | 1.2% / 4.6% | 1.2% / 4.9% |
| *Committee* error | 0.2% / 3.0% | 1.2% / 2.8% | 0.3% / 2.2% | 0.1% / 2.3% |

On average, Algorithm C in four-block mode worked about three ($\approx 63.2/22.4$)

times faster than online-BP, and thus can work faster than batch-mode nonlinear-CG algorithms, since, reported in [12], online-BP worked faster than nonlinear-CG. Here, we also tested the *committee* methods (see Chap. 8 in [13]) that merely combined all (equally-weighted) outputs of the ten MLPs, which were optimized independently in this experiment. The committee error was better than the average error, as expected. Intriguingly, our block-mode learning schemes introduced small (harmless) *bias*, improving the test-data performance; specifically, the two-block mode yielded the best test error rate 2.2% even with this simple committee method.

## 4 Conclusion and Future Directions

Pearlmutter's method can construct Krylov subspaces efficiently for implementing iterative batch- or block-mode learning. In our simulation examples, the simpler version of Pearlmutter's method (see Algorithms C and D) worked excellently. But it would be of interest to investigate other real-life large-scale problems to find out the strengths of the full-Hessian based methods (see Algorithm E) perhaps with a more elaborate preconditioner, which would be much more time-consuming per epoch but may reduce the total time dramatically; hence, need to deal with a delicate balancing act. Beside the simple committee method, it would be worth examining our algorithms for implementing other statistical learning methods (e.g., boosting) in conjunction with appropriate numerical linear algebra techniques. These are part of our overlay ambitious goal for attacking practical large-scale problems.

## Footnotes

*The floating-point operation counts were measured by using PAPI (Performance Application Programming Interface); see http://icl.cs.utk.edu/projects/papi/.

## References

[1] Shun-ichi Amari. *Natural gradient works efficiently in learning.* In *Neural Computation*, 10, pp. 251–276, 1998.

[2] A. R. Conn, N. I. M. Gould, and P. L. Toint. *Trust-Region Methods.* SIAM, 2000.

[3] James W. Demmel. *Applied Numerical Linear Algebra.* SIAM, 1997.

[4] J. E. Dennis, D. M. Gay, and R. E. Welsch. "An Adaptive Nonlinear Least-Squares Algorithm." In *ACM Trans. on Mathematical Software*, 7(3), pp. 348–368, 1981.

[5] R. A. Jacobs, M. I. Jordan, S. J. Nowlan and G. E. Hinton. "Adaptive Mixtures of Local Experts." In *Neural Computation*, pp. 79–87, Vol. 3, No. 1, 1991.

[6] Eiji Mizutani and James W. Demmel. "On structure-exploiting trust-region regularized nonlinear least squares algorithms for neural-network learning." In *International Journal of Neural Networks*. Elsevier Science, Vol. 16, pp. 745-753, 2003.

[7] Eiji Mizutani and James W. Demmel. "On separable nonlinear least squares algorithms for neuro-fuzzy modular network learning." In *Proceedings of the IEEE Int'l Joint Conf. on Neural Networks*, Vol.3, pp. 2399–2404, Honolulu USA, May, 2002. (Available at http://www.cs.berkeley.edu/~eiji/ijcnn02.pdf.)

[8] Eiji Mizutani and Stuart E. Dreyfus. "On complexity analysis of supervised MLP-learning for algorithmic comparisons." In *Proceedings of the INNS-IEEE Int'l Joint Conf. on Neural Networks*, Vol. 1, pp. 347–352, Washington D.C., July, 2001.

[9] Jorge J. Moré and Danny C. Sorensen. "Computing A Trust Region Step." In *SIAM J. Sci. Stat. Comp.* 4(3), pp. 553–572, 1983.

[10] Trond Steihaug "The Conjugate Gradient Method and Trust Regions in Large Scale Optimization." In *SIAM J. Numer. Anal.* pp. 626–637, vol. 20, no. 3. 1983.

[11] Barak A. Pearlmutter. "Fast exact multiplication by the Hessian." In *Neural Computation*, pp. 147–160, Vol. 6, No. 1, 1994.

[12] Holger Schwenk and Yoshua Bengio. "Boosting neural networks." In *Neural Computation*, pp. 1869–1887, Vol. 12, No. 8, 2000.

[13] Trevor Hastie, Robert Tibshirani, and Jerome Friedman. *The Elements of Statistical Learning.* Springer-Verlag, 2001 (Corrected printing 2002).
